# The Decision List Machine

**Marina Sokolova**
SITE, University of Ottawa
Ottawa, Ont. Canada,K1N-6N5
*sokolova@site.uottawa.ca*

**Mario Marchand**
SITE, University of Ottawa
Ottawa, Ont. Canada,K1N-6N5
*marchand@site.uottawa.ca*

**Nathalie Japkowicz**
SITE, University of Ottawa
Ottawa, Ont. Canada,K1N-6N5
*nat@site.uottawa.ca*

**John Shawe-Taylor**
Royal Holloway, University of London
Egham, UK, TW20-0EX
*jst@cs.rhul.ac.uk*

## Abstract

We introduce a new learning algorithm for decision lists to allow features that are constructed from the data and to allow a trade-off between accuracy and complexity. We bound its generalization error in terms of the number of errors and the size of the classifier it finds on the training data. We also compare its performance on some natural data sets with the set covering machine and the support vector machine.

## 1   Introduction

The set covering machine (SCM) has recently been proposed by Marchand and Shawe-Taylor (2001, 2002) as an alternative to the support vector machine (SVM) when the objective is to obtain a sparse classifier with good generalization. Given a feature space, the SCM tries to find the smallest conjunction (or disjunction) of features that gives a small training error. In contrast, the SVM tries to find the maximum soft-margin separating hyperplane on all the features. Hence, the two learning machines are fundamentally different in what they are trying to achieve on the training data.

To investigate if it is worthwhile to consider larger classes of functions than just the conjunctions and disjunctions that are used in the SCM, we focus here on the class of decision lists introduced by Rivest (1987) because this class strictly includes both conjunctions and disjunctions and is strictly included in the class of linear threshold functions (Marchand and Golea, 1993). Hence, we denote by *decision list machine* (DLM) any classifier which computes a decision list of Boolean-valued features, including features that are possibly constructed from the data. In this paper, we use the set of features introduced by Marchand and Shawe-Taylor (2001, 2002) known as data-dependent balls. By extending the sample compression technique of Littlestone and Warmuth (1986), we bound the generalization error of the DLM with data-dependent balls in terms of the number of errors and the number of balls it achieves on the training data. We also show that the DLM with balls can provide

better generalization than the SCM with this same set of features on some natural data sets.

## 2 The Decision List Machine

Let $\mathbf{x}$ denote an arbitrary $n$-dimensional vector of the input space $X$ which could be arbitrary subsets of $\Re^n$. We consider binary classification problems for which the training set $S = P \cup N$ consists of a set $P$ of positive training examples and a set $N$ of negative training examples. We define a *feature* as an arbitrary Boolean-valued function that maps $X$ onto $\{0, 1\}$. Given any set $\mathcal{H} = \{h_i(\mathbf{x})\}_{i=1}^{|\mathcal{H}|}$ of features $h_i(\mathbf{x})$ and any training set $S$, the learning algorithm returns a small subset $\mathcal{R} \subset \mathcal{H}$ of features. Given that subset $\mathcal{R}$, and an arbitrary input vector $\mathbf{x}$, the output $f(\mathbf{x})$ of the Decision List Machine (DLM) is defined to be:

If $(h_1(\mathbf{x}))$ then $b_1$

Else If $(h_2(\mathbf{x}))$ then $b_2$

...

Else If $(h_r(\mathbf{x}))$ then $b_r$

Else $b_{r+1}$

where each $b_i \in 0, 1$ defines the output of $f(\mathbf{x})$ if and only if $h_i$ is the first feature to be satisfied on $\mathbf{x}$ (*i.e.* the smallest $i$ for which $h_i(\mathbf{x}) = 1$). The constant $b_{r+1}$ (where $r = |\mathcal{R}|$) is known as the *default value*. Note that $f$ computes a disjunction of the $h_i$s whenever $b_i = 1$ for $i = 1 \ldots r$ and $b_{r+1} = 0$. To compute a conjunction of $h_i$s, we simply place in $f$ the negation of each $h_i$ with $b_i = 0$ for $i = 1 \ldots r$ and $b_{r+1} = 1$. Note, however, that a DLM $f$ that contains one or many *alternations* (*i.e.* a pair $(b_i, b_{i+1})$ for which $b_i \neq b_{i+1}$ for $i < r$) cannot be represented as a (pure) conjunction or disjunction of $h_i$s (and their negations). Hence, the class of decision lists strictly includes conjunctions and disjunctions.

From this definition, it seems natural to use the following greedy algorithm for building a DLM from a training set. For a given set $S' = P' \cup N'$ of examples (where $P' \subseteq P$ and $N' \subseteq N$) and a given set $\mathcal{H}$ of features, consider only the features $h_i \in \mathcal{H}$ which make no errors on either $P'$ or $N'$. If $h_i$ makes no error with $P'$, let $Q_i$ be the subset of examples of $N'$ on which $h_i$ makes no errors. Otherwise, if $h_i$ makes no error with $N'$, let $Q_i$ be the subset of examples of $P'$ on which $h_i$ makes no errors. In both cases we say that $h_i$ is *covering* $Q_i$. The greedy algorithm starts with $S' = S$ and an empty DLM. Then it finds the $h_i$ with the largest $|Q_i|$ and appends this $h_i$ to the DLM. It then removes $Q_i$ from $S'$ and repeat to find the $h_k$ with the largest $|Q_k|$ until either $P'$ or $N'$ is empty. It finally assigns $b_{r+1}$ to the class label of the remaining non-empty set.

Following Rivest (1987), this greedy algorithm is assured to build a DLM that makes no training errors whenever *there exists* a DLM on a set $\mathcal{E} \subseteq \mathcal{H}$ of features that makes zero training errors. However, this constraint is not really required in practice since we do want to permit the user of a learning algorithm to control the tradeoff between the accuracy achieved on the training data and the complexity (here the size) of the classifier. Indeed, a small DLM which makes a few errors on the training set might give better generalization than a larger DLM (with more features) which makes zero training errors. One way to include this flexibility is to early-stop the greedy algorithm when there remains a few more training examples to be covered. But a further reduction in the size of the DLM can be accomplished

**Algorithm BuildDLM$(P, N, p_p, p_n, s, \mathcal{H})$**

Input: A set $P$ of positive examples, a set $N$ of negative examples, the penalty values $p_p$ and $p_n$ , a stopping point $s$, and a set $\mathcal{H} = \{h_i(\mathbf{x})\}_{i=1}^{|\mathcal{H}|}$ of Boolean-valued features.

Output: A decision list $f$ consisting of a set $\mathcal{R} = \{(h_i, b_i)\}_{i=1}^{r}$ of features $h_i$ with their corresponding output values $b_i$, and a default value $b_{r+1}$.

Initialization: $\mathcal{R} = \emptyset, P' = P, N' = N$

1. For each $h_i \in \mathcal{H}$, let $P_i$ and $N_i$ be respectively the subsets of $P'$ and $N'$ correctly classified by $h_i$. For each $h_i$ compute $U_i$, where:

$$U_i \stackrel{\text{def}}{=} \max\{|P_i| - p_n \cdot |N' - N_i|,\ |N_i| - p_p \cdot |P' - P_i|\}$$

2. Let $h_k$ be a feature with the largest value of $U_k$.
3. If $(|P_k| - p_n \cdot |N' - N_k| \geq |N_k| - p_p \cdot |P' - P_k|)$ then $\mathcal{R} = \mathcal{R} \cup \{(h_k, 1)\}$, $P' = P' - P_k$, $N' = N_k$.
4. If $(|P_k| - p_n \cdot |N' - N_k| < |N_k| - p_p \cdot |P' - P_k|)$ then $\mathcal{R} = \mathcal{R} \cup \{(\neg h_k, 0)\}$, $N' = N' - N_k$, $P' = P_k$.
5. Let $r = |\mathcal{R}|$. If $(r < s$ and $P' \neq \emptyset$ and $N' \neq \emptyset)$ then go to step 1
6. Set $b_{r+1} = \neg b_r$. Return $f$.

Figure 1: The learning algorithm for the Decision List Machine

by considering features $h_i$ that do make a few errors on $P'$ (or $N'$) if many more examples $Q_i \in N'$ (or $Q_i \in P'$) can be covered.

Hence, to include this flexibility in choosing the proper tradeoff between complexity and accuracy, we propose the following modification of the greedy algorithm. For every feature $h_i$, let us denote by $P_i$ the subset of $P'$ on which $h_i$ makes no errors and by $N_i$ the subset of $N'$ on which $h_i$ makes no error. The above greedy algorithm is considering only features for which we have either $P_i = P'$ or $N_i = N'$, but to allow small deviation from these choices, we define the *usefullness* $U_i$ of feature $h_i$ by

$$U_i \stackrel{\text{def}}{=} \max\{|P_i| - p_n \cdot |N' - N_i|,\ |N_i| - p_p \cdot |P' - P_i|\}$$

where $p_n$ denotes the penalty of making an error on a negative example whereas $p_p$ denotes the penalty of making an error on a positive example.

Hence, each greedy step will be modified as follows. For a given set $S' = P' \cup N'$, we will select the feature $h_i$ with the largest value of $U_i$ and append this $h_i$ in the DLM. If $|P_i| - p_n \cdot |N' - N_i| \geq |N_i| - p_p \cdot |P' - P_i|$, we will then remove from $S'$ every example in $P_i$ (since they are correctly classified by the current DLM) *and* we will also remove from $S'$ every example in $N' - N_i$ (since a DLM with this feature is already misclassifying $N' - N_i$, and, consequently, the training error of the DLM will not increase if later features err on examples in $N' - N_i$). Otherwise if $|P_i| - p_n \cdot |N' - N_i| < |N_i| - p_p \cdot |P' - P_i|$, we will then remove from $S'$ examples in $N_i \cup (P' - P_i)$. Hence, we recover the simple greedy algorithm when $p_p = p_n = \infty$.

The formal description of our learning algorithm is presented in Figure 1. The penalty parameters $p_p$ and $p_n$ and the early stopping point $s$ are the model-selection parameters that give the user the ability to control the proper tradeoff between the training accuracy and the size of the DLM. Their values could be determined either

by using k-fold cross-validation, or by computing our bound (see section 4) on the generalization error. It therefore generalizes the learning algorithm of Rivest (1987) by providing this complexity-accuracy tradeoff and by permitting the use of any kind of Boolean-valued features, including those that are constructed from the data. Finally let us mention that Dhagat and Hellerstein (1994) did propose an algorithm for learning decision lists of few relevant attributes but this algorithm is not practical in the sense that it provides no tolerance to noise and does not easily accommodate parameters to provide a complexity-accuracy tradeoff.

## 3 Data-Dependent Balls

For each training example $\mathbf{x}_i$ with label $y_i \in \{0, 1\}$ and (real-valued) radius $\rho$, we define feature $h_{i,\rho}$ to be the following *data-dependent ball* centered on $\mathbf{x}_i$:

$$h_{i,\rho}(\mathbf{x}) \stackrel{\text{def}}{=} h_\rho(\mathbf{x}, \mathbf{x}_i) = \begin{cases} y_i & \text{if } d(\mathbf{x}, \mathbf{x}_i) \leq \rho \\ \overline{y}_i & \text{otherwise} \end{cases}$$

where $\overline{y}_i$ denotes the Boolean complement of $y_i$ and $d(\mathbf{x}, \mathbf{x}')$ denotes the distance between $\mathbf{x}$ and $\mathbf{x}'$. Note that any metric can be used for $d$. So far, we have used only the $L_1, L_2$ and $L_\infty$ metrics but it is certainly worthwhile to try to use metrics that actually incorporate some knowledge about the learning task. Moreover, we could use metrics that are obtained from the definition of an inner product $k(\mathbf{x}, \mathbf{x}')$.

Given a set $S$ of $m$ training examples, our initial set of features consists, in principle, of $\mathcal{H} = \bigcup_{i \in S} \bigcup_{\rho \in [0, \infty[} h_{i,\rho}$. But obviously, for each training example $\mathbf{x}_i$, we need only to consider the set of $m - 1$ distances $\{d(\mathbf{x}_i, \mathbf{x}_j)\}_{j \neq i}$. This reduces our initial set $\mathcal{H}$ to $O(m^2)$ features. In fact, from the description of the DLM in the previous section, it follows that the ball with the largest usefulness belongs to one of the following following types of balls: type $\mathcal{P}_i$, $\mathcal{P}_o$, $\mathcal{N}_i$, and $\mathcal{N}_o$.

Balls of type $\mathcal{P}_i$ (positive inside) are balls having a positive example $\mathbf{x}$ for its *center* and a radius given by $\rho = d(\mathbf{x}, \mathbf{x}') - \epsilon$ for some negative example $\mathbf{x}'$ (that we call a *border point*) and very small positive number $\epsilon$. Balls of type $\mathcal{P}_o$ (positive outside) have a negative example center $\mathbf{x}$ and a radius $\rho = d(\mathbf{x}, \mathbf{x}') + \epsilon$ given by a negative border $\mathbf{x}'$. Balls of type $\mathcal{N}_i$ (negative inside) have a negative center $\mathbf{x}$ and a radius $\rho = d(\mathbf{x}, \mathbf{x}') - \epsilon$ given by a positive border $\mathbf{x}'$. Balls of type $\mathcal{N}_o$ (negative outside) have a positive center $\mathbf{x}$ and a radius $\rho = d(\mathbf{x}, \mathbf{x}') + \epsilon$ given by a positive border $\mathbf{x}'$.

This proposed set of features, constructed from the training data, provides to the user full control for choosing the proper tradeoff between training accuracy and function size.

## 4 Bound on the Generalization Error

Note that we cannot use the "standard" VC theory to bound the expected loss of DLMs with data-dependent features because the VC dimension is a property of a function class defined on some input domain *without reference* to the data. Hence, we propose another approach.

Since our learning algorithm tries to build a DLM with the smallest number of data-dependent balls, we seek a bound that depends on this number and, consequently, on the number of examples that are used in the final classifier (the hypothesis). We can thus think of our learning algorithm as compressing the training set into a small subset of examples that we call the *compression set*. It was shown by Littlestone and Warmuth (1986) and Floyd and Warmuth (1995) that we can bound

the generalization error of the hypothesis $f$ if we can always reconstruct $f$ from the compression set. Hence, the only requirement is the existence of such a *reconstruction function* and its only purpose is to permit the exact identification of the hypothesis from the compression set and, possibly, additional bits of information. Not surprisingly, the bound on the generalization error increases rapidly in terms of these additional bits of information. So we must make minimal usage of them.

We now describe our reconstruction function and the additional information that it needs to assure, in all cases, the proper reconstruction of the hypothesis from a compression set. Our proposed scheme works in all cases provided that the learning algorithm returns a hypothesis that always correctly classifies the compression set (but not necessarily all of the training set). Hence, we need to add this constraint in **BuildDLM** for our bound to be valid but, in practice, we have not seen any significant performance variation introduced by this constraint. We first describe the simpler case where only balls of types $\mathcal{P}_i$ and $\mathcal{N}_i$ are permitted and, later, describe the additional requirements that are introduced when we also permit balls of types $\mathcal{P}_o$ and $\mathcal{N}_o$.

Given a compression set $\Lambda$ (returned by the learning algorithm), we first partition it into four disjoint subsets $C_p, C_n, B_p$, and $B_n$ consisting of positive ball centers, negative ball centers, positive borders, and negative borders respectively. Each example in $\Lambda$ is specified only once. When only balls of type $\mathcal{P}_i$ and $\mathcal{N}_i$ are permitted, the center of a ball cannot be the center of another ball since the center is removed from the remaining examples to be covered when a ball is added to the DLM. But a center can be the border of a previous ball in the DLM and a border can be the border of more than one ball. Hence, points in $B_p \cup B_n$ are examples that are borders without being the center of another ball. Because of the crucial importance of the ordering of the features in a decision list, these sets do not provide enough information by themselves to be able to reconstruct the hypothesis. To specify the ordering of *each* ball center it is sufficient to provide $\log_2(r)$ bits of additional information where the number $r$ of balls is given by $r = c_p + c_n$ for $c_p = |C_p|$ and $c_n = |C_n|$. To find the radius $\rho_i$ for each center $\mathbf{x}_i$ we start with $C_p' = C_p, C_n' = C_n, B_p' = B_p, B_n' = B_n$, and do the following, sequentially from the first center to the last. If center $\mathbf{x}_i \in C_p'$, then the radius is given by $\rho_i = \min_{\mathbf{x}_j \in C_n' \cup B_n'} d(\mathbf{x}_i, \mathbf{x}_j) - \epsilon$ and we remove center $\mathbf{x}_i$ from $C_p'$ and any other point from $B_p'$ covered by this ball (to find the radius of the other balls). If center $\mathbf{x}_i \in C_n'$, then the radius is given by $\rho_i = \min_{\mathbf{x}_j \in C_p' \cup B_p'} d(\mathbf{x}_i, \mathbf{x}_j) - \epsilon$ and we remove center $\mathbf{x}_i$ from $C_n'$ and any other point from $B_n'$ covered by this ball. The output $b_i$ for each ball $h_i$ is 1 if the center $\mathbf{x}_i \in C_p$ and 0 otherwise. This reconstructed decision list of balls will be the same as the hypothesis if and only if the compression set is always correctly classified by the learning algorithm. Once we can identify the hypothesis from the compression set, we can bound its generalization error.

**Theorem 1** *Let $S = P \cup N$ be a training set of positive and negative examples of size $m = m_p + m_n$. Let $A$ be the learning algorithm* **BuildDLM** *that uses data-dependent balls of type $\mathcal{P}_i$ and $\mathcal{N}_i$ for its set of features with the constraint that the returned function $A(S)$ always correctly classifies every example in the compression set. Suppose that $A(S)$ contains $r$ balls, and makes $k_p$ training errors on $P$, $k_n$ training errors on $N$ (with $k = k_p + k_n$), and has a compression set $\Lambda = C_p \cup C_n \cup B_p \cup B_n$ (as defined above) of size $\lambda = c_p + c_n + b_p + b_n$ . With probability $1 - \delta$ over all random training sets $S$ of size $m$, the generalization error $\mathrm{er}(A(S))$ of $A(S)$ is bounded by*

$$\mathrm{er}(A(S)) \quad \leq \quad 1 - \exp\left\{ \frac{-1}{m - \lambda - k} \left( \ln B_\lambda + \ln(r!) + \ln \frac{1}{\delta_\lambda} \right) \right\}$$

*where* $\delta_\lambda \overset{\text{def}}{=} \left(\frac{\pi^2}{6}\right)^{-6} \cdot ((c_p+1)(c_n+1)(b_p+1)(b_n+1)(k_p+1)(k_n+1))^{-2} \cdot \delta$  *and where*

$$B_\lambda \overset{\text{def}}{=} \binom{m_p}{c_p}\binom{m_p - c_p}{b_p}\binom{m_n}{c_n}\binom{m_n - c_n}{b_n}\binom{m_p - c_p - b_p}{k_p}\binom{m_n - c_n - b_n}{k_n}$$

**Proof** Let $\mathcal{X}$ be the set of training sets of size $m$. Let us first bound the probability $P_{\mathbf{m}} \overset{\text{def}}{=} P\{S \in \mathcal{X} : \mathrm{er}(A(S)) \geq \epsilon \mid \mathbf{m}(S) = \mathbf{m}\}$ given that $\mathbf{m}(S)$ is fixed to some value $\mathbf{m}$ where $\mathbf{m} \overset{\text{def}}{=} (m, m_p, m_n, c_p, c_n, b_p, b_n, k_p, k_n)$. For this, denote by $\mathcal{E}_p$ the subset of $P$ on which $A(S)$ makes an error and similarly for $\mathcal{E}_n$. Let $I$ be the message of $\log_2(r!)$ bits needed to specify the ordering of the balls (as described above). Now define $P'_{\mathbf{m}}$ to be

$$P'_{\mathbf{m}} \overset{\text{def}}{=} P\{S \in \mathcal{X} : \mathrm{er}(A(S)) \geq \epsilon \mid C_p = S_1, C_n = S_2, B_p = S_3, B_n = S_4$$
$$\mathcal{E}_p = S_5, \mathcal{E}_n = S_6, I = I_0, \mathbf{m}(S) = \mathbf{m}\}$$

for some fixed set of disjoint subsets $\{S_i\}_{i=1}^6$ of $S$ and some fixed information message $I_0$. Since $B_\lambda$ is the number of different ways of choosing the different compression subsets and set of error points in a training set of fixed $\mathbf{m}$, we have:

$$P_{\mathbf{m}} \leq (r!) \cdot B_\lambda \cdot P'_{\mathbf{m}}$$

where the first factor comes from the additional information that is needed to specify the ordering of $r$ balls. Note that the hypothesis $f \overset{\text{def}}{=} A(S)$ is fixed in $P'_{\mathbf{m}}$ (because the compression set is fixed and the required information bits are given). To bound $P'_{\mathbf{m}}$, we make the standard assumption that each example $\mathbf{x}$ is independently and identically generated according to some fixed but unknown distribution. Let $p$ be the probability of obtaining a positive example, let $\alpha$ be the probability that the fixed hypothesis $f$ makes an error on a positive example, and let $\beta$ be the probability that $f$ makes an error on a negative example. Let $t_p \overset{\text{def}}{=} c_p + b_p + k_p$ and let $t_n \overset{\text{def}}{=} c_n + b_n + k_n$. We then have:

$$
\begin{aligned}
P'_{\mathbf{m}} &= (1-\alpha)^{m_p - t_p}(1-\beta)^{m - t_n - m_p}\binom{m - t_n - t_p}{m_p - t_p}p^{m_p - t_p}(1-p)^{m - t_n - m_p} \\
&\leq \sum_{m' = t_p}^{m - t_n}(1-\alpha)^{m' - t_p}(1-\beta)^{m - t_n - m'}\binom{m - t_n - t_p}{m' - t_p}p^{m' - t_p}(1-p)^{m - t_n - m'} \\
&= [(1-\alpha)p + (1-\beta)(1-p)]^{m - t_n - t_p} = (1 - \mathrm{er}(f))^{m - t_n - t_p} \\
&\leq (1 - \epsilon)^{m - t_n - t_p}
\end{aligned}
$$

Consequently:

$$P_{\mathbf{m}} \leq (r!) \cdot B_\lambda \cdot (1 - \epsilon)^{m - t_n - t_p}.$$

The theorem is obtained by bounding this last expression by the proposed value for $\delta_\lambda(\mathbf{m})$ and solving for $\epsilon$ since, in that case, we satisfy the requirement that

$$P\left\{S \in \mathcal{X} : \mathrm{er}(A(S)) \geq \epsilon\right\} = \sum_{\mathbf{m}} P_{\mathbf{m}} P\left\{S \in \mathcal{X} : \mathbf{m}(S) = \mathbf{m}\right\}$$

$$\leq \sum_{\mathbf{m}} \delta_\lambda(\mathbf{m}) P\left\{S \in \mathcal{X} : \mathbf{m}(S) = \mathbf{m}\right\} \leq \sum_{\mathbf{m}} \delta_\lambda(\mathbf{m}) = \delta$$

where the sums are over all possible realizations of $\mathbf{m}$ for a fixed $m_p$ and $m_n$. With the proposed value for $\delta_\lambda(\mathbf{m})$, the last equality follows from the fact that

$\sum_{i=1}^{\infty}(1/i^2) = \pi^2/6$. ∎

The use of balls of type $\mathcal{P}_o$ and $\mathcal{N}_o$ introduces a few more difficulties that are taken into account by sending more bits to the reconstruction function. First, the center of a ball of type $\mathcal{P}_o$ and $\mathcal{N}_o$ can be used for more than one ball since the covered examples are outside the ball. Hence, the number $r$ of balls can now exceed $c_p + c_n = c$. So, to specify $r$, we can send $\log_2(\lambda)$ bits. Then, for each ball, we can send $\log_2 c$ bits to specify which center this ball is using and another bit to specify if the examples covered are inside or outside the ball. Using the same notation as before, the radius $\rho_i$ of a center $\mathbf{x}_i$ of a ball of type $\mathcal{P}_o$ is given by $\rho_i = \max_{\mathbf{x}_j \in C_n' \cup B_n'} d(\mathbf{x}_i, \mathbf{x}_j) + \epsilon$, and for a center $\mathbf{x}_i$ of a ball of type $\mathcal{N}_o$, its radius is given by $\rho_i = \max_{\mathbf{x}_j \in C_p' \cup B_p'} d(\mathbf{x}_i, \mathbf{x}_j) + \epsilon$. With these modifications, the same proof of Theorem 1 can be used to obtain the next theorem.

**Theorem 2** *Let A be the learning algorithm* **BuildDLM** *that uses data-dependent balls of type $\mathcal{P}_i$, $\mathcal{N}_i$, $\mathcal{P}_o$, and $\mathcal{N}_o$ for its set of features. Consider all the definitions used for Theorem 1 with $c \stackrel{\text{def}}{=} c_p + c_n$. With probability $1 - \delta$ over all random training sets S of size m, we have*

$$
\text{er}(A(S)) \quad \leq \quad 1 - \exp\left\{\frac{-1}{m - \lambda - k}\left(\ln B_\lambda + \ln \lambda + r\ln(2c) + \ln\frac{1}{\delta_\lambda}\right)\right\}
$$

Basically, our bound states that good generalization is expected when we can find a small DLM that makes few training errors. In principle, we could use it as a guide for choosing the model selection parameters $s$, $p_p$, and $p_n$ since it depends only on what the hypothesis has achieved on the training data.

## 5  Empirical Results on Natural data

We have compared the practical performance of the DLM with the support vector machine (SVM) equipped with a Radial Basis Function kernel of variance $1/\gamma$. The data sets used and the results obtained are reported in Table 1. All these data sets where obtained from the machine learning repository at UCI. For each data set, we have removed all examples that contained attributes with unknown values (this has reduced substantially the "votes" data set) and we have removed examples with contradictory labels (this occurred only for a few examples in the Haberman data set). The remaining number of examples for each data set is reported in Table 1. No other preprocessing of the data (such as scaling) was performed. For all these data sets, we have used the 10-fold cross validation error as an estimate of the generalization error. The values reported are expressed as the total number of errors (*i.e.* the sum of errors over all testing sets). We have ensured that each training set and each testing set, used in the 10-fold cross validation process, was the same for each learning machine (*i.e.* each machine was trained on the same training sets and tested on the same testing sets).

The results reported for the SVM are only those obtained for the best values of the kernel parameter $\gamma$ and the soft margin parameter $C$ found among an *exhaustive* list of *many* values. The values of these parameters are reported in Marchand and Shawe-Taylor (2002). The "size" column refers to the average number of support vectors contained in SVM machines obtained from the 10 different training sets of 10-fold cross-validation.

We have reported the results for the SCM (Marchand and Shawe-Taylor, 2002) and the DLM when both machines are equipped with data-dependent balls under the $L_2$ metric. For the SCM, the $T$ column refers to type of the best machine found

| Data Set | | SVM | | SCM with balls | | | | DLM with balls | | | | |
|---|---|---|---|---|---|---|---|---|---|---|---|---|
| Name | #exs | size | errors | $T$ | $p$ | $s$ | errors | $T$ | $p_p$ | $p_n$ | $s$ | errors |
| BreastW | 683 | 58 | 19 | c | 1.8 | 2 | 15 | c | 2.1 | 1 | 2 | **14** |
| Votes | 52 | 18 | **3** | d | 0.9 | 1 | 6 | s | 0.1 | 0.3 | 1 | **3** |
| Pima | 768 | 526 | 203 | c | 1.1 | 3 | **189** | c | 1.5 | 1.5 | 6 | **189** |
| Haberman | 294 | 146 | 71 | c | 1.4 | 1 | 71 | s | 2 | 3 | 7 | **65** |
| Bupa | 345 | 266 | 107 | d | 2.8 | 9 | **106** | c | 2 | 2 | 4 | 108 |
| Glass | 214 | 125 | 34 | d | $\infty$ | 2 | 36 | c | 4.8 | $\infty$ | 12 | **28** |
| Credit | 653 | 423 | **190** | d | 1.2 | 4 | 194 | c | 1 | $\infty$ | 11 | 197 |

Table 1: Data sets and results for SVMs, SCMs, and DLMs.

($c$ for conjunction, and $d$ for disjunction), the $p$ column refers the best value found for the penalty parameter, and the $s$ column refers the the best stopping point in terms of the number of balls. The same definitions applies also for DLMs except that two different penalty values ($p_p$ and $p_n$) are used. In the $T$ column of the DLM results, we have specified by s (simple) when the DLM was trained by using only balls of type $\mathcal{P}_i$ and $\mathcal{N}_i$ and by c (complex) when the four possible types of balls where used (see section 3). Again, only the values that gave the smallest 10-fold cross-validation error are reported.

The most striking feature in Table 1 is the level of sparsity achieved by the SCM and the DLM in comparison with the SVM. This difference is huge. The other important feature is that DLMs often provide slightly better generalization than SCMs and SVMs. Hence, DLMs can provide a good alternative to SCMs and SVMs.

**Acknowledgments**

Work supported by NSERC grant OGP0122405 and, in part, by the EU under the NeuroCOLT2 Working Group, No EP 27150.

# References

Aditi Dhagat and Lisa Hellerstein. PAC learning with irrelevant attributes. In *Proc. of the 35rd Annual Symposium on Foundations of Computer Science*, pages 64–74. IEEE Computer Society Press, Los Alamitos, CA, 1994.

Sally Floyd and Manfred Warmuth. Sample compression, learnability, and the Vapnik-Chervonenkis dimension. *Machine Learning*, 21(3):269–304, 1995.

N. Littlestone and M. Warmuth. Relating data compression and learnability. Technical report, University of California Santa Cruz, 1986.

Mario Marchand and Mostefa Golea. On learning simple neural concepts: from halfspace intersections to neural decision lists. *Network: Computation in Neural Systems*, 4:67–85, 1993.

Mario Marchand and John Shawe-Taylor. Learning with the set covering machine. *Proceedings of the Eighteenth International Conference on Machine Learning (ICML 2001)*, pages 345–352, 2001.

Mario Marchand and John Shawe-Taylor. The set covering machine. *Journal of Machine Learning Reasearch (to appear)*, 2002.

Ronald L. Rivest. Learning decision lists. *Machine Learning*, 2:229–246, 1987.
